# Intransitive Likelihood-Ratio Classifiers

**Jeff Bilmes**    *and*    **Gang Ji**
Department of Electrical Engineering
University of Washington
Seattle, WA 98195-2500
{*bilmes,gji*}*@ee.washington.edu*

**Marina Meilă**
Department of Statistics
University of Washington
Seattle, WA 98195-4322
*mmp@stat.washington.edu*

## Abstract

In this work, we introduce an information-theoretic based correction term to the likelihood ratio classification method for multiple classes. Under certain conditions, the term is sufficient for optimally correcting the difference between the true and estimated likelihood ratio, and we analyze this in the Gaussian case. We find that the new correction term significantly improves the classification results when tested on medium vocabulary speech recognition tasks. Moreover, the addition of this term makes the class comparisons analogous to an intransitive game and we therefore use several tournament-like strategies to deal with this issue. We find that further small improvements are obtained by using an appropriate tournament. Lastly, we find that intransitivity appears to be a good measure of classification confidence.

## 1  Introduction

An important aspect of decision theory is multi-way pattern classification whereby one must determine the class $c^*$ for a given data vector $x$ that minimizes the overall risk:

$$c^* = \operatorname*{argmin}_{c'} \sum_c L(c'|c)p(c|x),$$

where $L(c'|c)$ is the loss in choosing $c'$ when the true class is $c$. This decision rule is provably optimal for the given loss function [3]. For the 0/1-loss functions, it is optimal to simply use the posterior probability to determine the optimal class

$$c^* = \operatorname*{argmax}_c p(c|x)$$

This procedure may equivalently be specified using a tournament style game-playing strategy. In this case, there is an implicit class ordering $\{c_1, c_2, ..., c_N\}$, and a class-pair ($i$ and $j$) scoring function for an unknown sample $x$:

$$S_{ij}(x) = L_{ij}(x) + R_{ij}$$

such that $L_{ij} = \log p(x|i)/p(x|j)$ is the log-likelihood ratio and $R_{ij} = \log p(i)/p(j)$ is the log prior odds. The strategy proceeds by evaluating $S_{c_1 c_2}$ which if positive is followed by $S_{c_1 c_3}$ and otherwise by $S_{c_3 c_2}$. This continues until a "winner" is found. Of course, the order of the classes does not matter, as the same winner is found for all permutations. In

any event, this style of classification can be seen as a transitive game [5] between players who correspond to the individual classes.

In this work we extend the likelihood-ratio based classification with a term, based on the Kullback-Leibler divergence [2], that expresses the inherent posterior confusability between the underlying likelihoods being compared for a given pair of players. We find that by including this term, the results of a classification system significantly improve, without changing or increasing the quantity of the estimated free model parameters. We also show how, under certain assumptions, the term can be seen as an optimal correction between the estimated model likelihood ratio and the true likelihood ratio, and gain further intuition by examining the case when the likelihoods $p(x|i)$ are Gaussians. Furthermore, we observe that the new strategy leads to an intransitive game [5], and we investigate several strategies for playing such games. This results in further (but small) improvements. Finally, we consider the instance of intransitivity as a confidence measure, and investigate an iterative approach to further improve the correction term.

Section 2 first motivates and defines our approach, and shows the conditions under which it is optimal. Section 2.1 then reports experimental results which show significant improvements where the likelihoods are hidden Markov models trained on speech data. Section 3 then recasts the procedure as intransitive games, and evaluates a variety of game playing strategies yielding further (small) error reductions. Section 3.1 attempts to better understand our results via empirical analysis, and evaluates additional classification strategies. Section 4 explores an iterative strategy for improving our technique, and finally Section 5 concludes and discusses future work.

## 2  Extended Likelihood-Ratio-based Classification

The Kullback-Leibler (KL) divergence[2], an asymmetric measure of the distance between two probability densities, is defined as follows:

$$D(p\|q) = E_p \log \frac{p(X)}{q(X)}$$

where $p$ and $q$ are probability densities over the same sample space. The KL-divergence is also called the average (under $p$) information for discrimination in favor of $p$ over $q$. For our purposes, we are interested in KL-divergence between class-conditional likelihoods $p(x|i)$ where $i$ is the class number:

$$D(i\|j) = \int \log \frac{p(x|i)}{p(x|j)} p(x|i) dx$$

One intuitive way of viewing $D(i\|j)$ is as follows: if $D(i\|j)$ is small, then samples of class $i$ are more likely to be erroneously classified as class $j$ than when $D(i\|j)$ is large. Comparing $D(i\|j)$ and $D(j\|i)$ should tell us which of $i$ and $j$ is more likely to have its samples mis-classified by the other model. Therefore, the difference $D(i\|j) - D(j\|i)$, when positive, indicates that samples of class $j$ are more likely to be mis-classified as class $i$ than samples of class $i$ are to be mis-classified as class $j$ (and vice-versa when the difference is negative). In other words, $i$ "steals" from $j$ more than $j$ steals from $i$ when the difference is positive, thereby suggesting that class $j$ should receive aid in this case. This difference can be viewed as a form of posterior (i.e., based on the data) "bias" indicating which class should receive favor over the other.[1] We can adjust $L_{ij}$ (the log-likelihood ratio) with this posterior bias, to obtain a new function comparing classes $i$ and $j$ as follows:

$$S_{ij} = L_{ij} + R_{ij} - D_{ij}.$$

[1]Note that this is not the normal notion of statistical bias as in $E\Theta$ where $\Theta$ is an estimate of model parameters.

where

$$D_{ij} \triangleq \frac{1}{2} \left( D(i\|j) - D(j\|i) \right)$$

The likelihood ratio is adjusted in favor of $j$ when $D_{ij}$ is positive, and in favor of $i$ when $D_{ij}$ is negative. We then use $S_{ij}$, and when it is positive, choose class $i$.

The above intuition does not explain why such a correction factor should be used, since using $L_{ij}$ along with $R_{ij}$ is already optimal. In practice, however, we do not have access to the true likelihood ratios but instead to an approximation that has been estimated from training data. Let the variable $L_{ij}(x) = \log p(x|i)/p(x|j)$ be the true log-likelihood ratio, and $\hat{L}_{ij}(x) = \log \hat{p}(x|i)/\hat{p}(x|j)$ be the model-based log ratio. Furthermore, let

$$\hat{D}(i\|j) \triangleq \int \log \frac{\hat{p}(x|i)}{\hat{p}(x|j)} p(x|i) dx$$

be the modified KL-divergence between the class conditional models, measured modulo the true distribution $p(x|i)$, and let $\hat{D}_{ij} \triangleq \frac{1}{2} \left( \hat{D}(i\|j) - \hat{D}(j\|i) \right)$. Finally, let $R_{ij}$ (resp. $\hat{R}_{ij}$) be the true (resp. estimated) log prior odds. Our (usable) scoring function becomes:

$$S_{ij}(x) = \hat{L}_{ij}(x) + \hat{R}_{ij} - \hat{D}_{ij}. \tag{1}$$

which has an intuitive explanation similar to the above.

There are certain conditions under which the above approach is theoretically justifiable. Let us assume for now a two-class problem where $i$ and $j$ are the two classes, so $p(i)+p(j) = 1$. A sufficient condition for the estimated quantities above to yield optimal performance is for $L + R = \hat{L} + \hat{R}$ for all $x$.[2] Since this is not the case in practice, an $ij$-dependent constant term $\alpha$ may be added correcting for any differences as best as possible. This yields $L + R = \hat{L} + \hat{R} + \alpha$. We can define an $\alpha$-dependent cost function

$$\mathcal{L}(\alpha) = \int p(x) \left( L + R - \hat{L} - \hat{R} - \alpha \right)^2 dx$$

which, when minimized, yields $\alpha^* = E_{p(X)} L + R - E_{p(X)} \hat{L} - \hat{R}$ stating that the optimal $\alpha$ under this cost function is just the mean of the difference of the remaining terms. Note that $E_{p(X)} L = p(i) D(i\|j) - p(j) D(j\|i)$ and $E_{p(X)} \hat{L} = p(i) \hat{D}(i\|j) - p(j) \hat{D}(j\|i)$. Several additional assumptions lead to Equation 1. First, let us assume that the prior probabilities are equal (so $p(i) = 0.5$) and that the estimated and true priors are negligibly different (i.e., $R - \hat{R} \approx 0$). Secondly, if we assume that $E_{p(X)} L = 0$, this implies that

$$p(i)/p(j) = D(j\|i)/D(i\|j)$$

which means that $D(j\|i) = D(i\|j)$ under equal priors. While KL-divergence is not symmetric in general, we can see that if this holds (or is approximately true for a given problem) then the remaining correction is $-E\hat{Y}$ exactly yielding $\hat{D}_{ij}$ in Equation 1.

To gain further insight, we can examine the case when the likelihoods are Gaussian univariate distributions, with means $\mu_i, \mu_j$ and variances $\sigma_i^2, \sigma_j^2$. In this case,

$$D_{ij} = \frac{1}{4} \left[ \frac{\sigma_i^2}{\sigma_j^2} - \frac{\sigma_j^2}{\sigma_i^2} + (\mu_i - \mu_j)^2 \left( \frac{1}{\sigma_j^2} - \frac{1}{\sigma_i^2} \right) \right] \tag{2}$$

It is easy to see that for $\sigma_i^2 = \sigma_j^2$ the value of $D_{ij}$ is zero for any $\mu_i, \mu_j$. By computing the derivative $\frac{\partial D_{ij}}{\partial \sigma_i^2}$ we can show that $D_{ij}$ is monotonically increasing with $\sigma_i^2$. Hence, $D_{ij}$ is positive iff $\sigma_i^2 \geq \sigma_j^2$ and therefore it penalizes the distribution (class) with higher variance.

| VOCAB SIZE | WER $L_{ij}$ | WER $S_{ij}$ |
|---|---|---|
| 75 | 2.33584 | 1.91561 |
| 150 | 3.31072 | 2.89833 |
| 300 | 5.22513 | 4.51365 |
| 600 | 7.39268 | 6.18517 |

Table 1: Word error rates (WER) for likelihood ratio $L_{ij}$ and augmented likelihood ratio $S_{ij}$ based classification for various numbers of classes (VOCAB SIZE).

Similar relations hold for multivariate Gaussians with means $\mu_i, \mu_j$ and variances $\Sigma_i, \Sigma_j$.

$$4D_{ij} = \text{tr}(\Sigma_i \Sigma_j^{-1} - \Sigma_j \Sigma_i^{-1}) + (\mu_i - \mu_j)^T (\Sigma_j^{-1} - \Sigma_i^{-1})(\mu_i - \mu_j) \tag{3}$$

The above is zero when the two covariance matrices are equal. This implies that for Gaussians with equal covariance matrices, $D(j\|i) = D(i\|j)$ and our correction term is optimal. This is the same as the condition for Fisher's linear discriminant analysis (LDA). Moreover, in the case $\Sigma_i = \alpha \Sigma_j$ with $\alpha > 0$, we have that $D_{ij} < 0$ for $\alpha < 1$ and $D_{ij} > 0$ for $\alpha > 1$ which again implies that $D_{ij}$ penalizes the class that has larger covariance.

## 2.1  Results

We tried this method (assuming that $R_{ij} = \hat{R}_{ij} = 0$) on a medium vocabulary speech recognition task. In our case the likelihood functions $p(x|i)$ are hidden Markov model (HMM) scores[3]. The task we chose is NYNEX PHONEBOOK[4], an isolated word speech corpus. Details of the experimental setup, training/test sets, and model topologies, are described in [1][4].

In general, there are a number of ways to compute $D_{ij}$. These include 1) analytically, using estimated model parameters (possible, for example, with Gaussian densities), 2) computing the KL-divergences on training data using a law-of-large-numbers-like average of likelihood ratios and using training-data estimated model parameters, 3) doing the same as 2 but using test data where hypothesized answers come from a first pass $L_{ij}$-based classification, and 4) Monte-Carlo methods where again the same procedure as 2 is used, but the data is sampled from the training-data estimated distributions. For HMMs, method 1 above is not possible. Also, the data set we used (PHONEBOOK) uses different classes for the training and test sets. In other words, the training and test vocabularies are different. During training, phone models are constructed that are pieced together for the test vocabularies. Therefore, method 2 above is also not possible for this data.

Either method 3 or 4 can be used in our case, and we used method 3 in all our experiments. Of course, using the true test labels in method 3 would be the ideal measure of the degree of confusion between models, but these are of course not available (see Figure 2, however, showing the results of a cheating experiment). Therefore, we use the hypothesized labels from a first stage to compute $\hat{D}_{ij}$.

The procedure thus is as follows: 1) obtain $p(x|i)$ using maximum likelihood EM training, 2) classify the test set using only $\hat{L}_{ij}$ and record the error rate, 3) using the hypothesized class labels (answers with errors) to step 2, compute $\hat{D}_{ij}$, 4) re-classify the test set using the score $S_{ij} = \hat{L}_{ij} - \hat{D}_{ij}$ and record the new error rate. $S_{ij}$ is used if either one of $D(i\|j)$

| VOCAB | $L_{ij}$ | RAND1 | RAND500 | RAND1000 | WORLD CUP |
|---|---|---|---|---|---|
| 75 | 2.33584 | 1.87198 | 1.82047 | 1.91467 | 2.12777 |
| 150 | 3.31072 | 2.88505 | 2.71881 | 2.72809 | 2.79516 |
| 300 | 5.22513 | 4.41428 | 4.34608 | 4.28930 | 3.81583 |
| 600 | 7.39268 | 6.15828 | 6.13085 | 5.91440 | 5.93883 |

Table 2: The WER under different tournament strategies

or $D(j\|i)$ is below a threshold (i.e., when a likely confusion exists), otherwise $\hat{L}_{ij}$ is used for classification.

Table 1 shows the result of this experiment. The first column shows the vocabulary size of the system (identical to the number of classes)[5]. The second column shows the word error rate (WER) using just $L_{ij}$, and the third column shows WER using $S_{ij}$. As can be seen, the WER decreases significantly with this approach. Note also that no additional free parameters are used to obtain these improvements.

## 3  Playing Games

We may view either $L_{ij}$ or $S_{ij}$ as providing a score of class $i$ over $j$ — when positive, class $i$ wins, and when negative, class $j$ wins. In general, the classification procedure may be viewed as a tournament-style game, where for a given sample $x$, different classes correspond to different players. Players pair together and play each other, and the winner goes on to play another match with a different player. The strategy leading to table 1 required a particular class presentation order — in that case the order was just the numeric ordering of the arbitrarily assigned integer classes (corresponding to words in this case).

Of course when $L_{ij}$ alone is used, the order of the comparisons do not matter, leading to a transitive game [5] (the order of player pairings do not change the final winner). The quantity $S_{ij}$, however, is not guaranteed to be transitive, and when used in a tournament it results in what is called an intransitive game[5]. This means, for example, that $A$ might win over $B$ who might win over $C$ who then might win over $A$. Games may be depicted as directed graphs, where an edge between two players point towards the winner. In an intransitive game, the graph contains directed cycles. There has been very little research on intransitive game strategies — there are in fact a number of philosophical issues relating to if such games are valid or truly exist. Nevertheless, we derived a number of tournament strategies for playing such intransitive games and evaluated their performance in the following.

Broadly, there are two tournament types that we considered. Given a particular ordering of the classes $\{c_1, c_2, ..., c_N\}$, we define a *sequential* tournament when $c_1$ plays $c_2$, the winner plays $c_3$, the winner plays $c_4$ and so on. We also define a *tree-based* tournament when $c_1$ plays $c_2$, $c_3$ plays $c_4$, and so on. The tree-based tournament is then applied recursively on the resulting $N/2$ winners until a final winner is found.

Based on the above, we investigated several intransitive game playing strategies. For RAND1, we just choose a single random tournament order in a sequential tournament. For RAND500, we run 500 sequential tournaments, each one with a different random order. The ultimate winner is taken to be the player who wins the most tournaments. The third strategy plays 1000 rather than 500 tournaments. The final strategy is inspired by world-cup soccer tournaments: given a randomly generated permutation, the class sequence is

| vocabulary | $2^H$ | var | max | $2^H$ | var | max |
|---|---|---|---|---|---|---|
| 75 | 1.0047 | 0.0071 | 2.7662 | 1.0285 | 0.0759 | 3.8230 |
| 150 | 1.0061 | 0.0126 | 3.6539 | 1.0118 | 0.0263 | 3.8724 |
| 300 | 1.0241 | 0.0551 | 4.0918 | 1.0170 | 0.0380 | 3.9072 |
| 600 | 1.0319 | 0.0770 | 5.0460 | 1.0533 | 0.1482 | 5.5796 |

Table 3: The statistics of winners. Columns 2-4: 500 random tournaments, Columns 5-7: 1000 random tournaments.

separated into 8 groups. We pick the winner of each group using a sequential tournament (the "regionals"). Then a tree-based tournament is used on the group winners.

Table 1 compares these different strategies. As can be seen, the results get slightly better (particularly with a larger number of classes) as the number of tournaments increases. Finally, the single word cup strategy does surprisingly well for the larger class sizes. Note that the improvements are statistically significant over the baseline (0.002 using a difference of proportions significance test) and the improvements are more dramatic for increasing vocabulary size. Furthermore, the it appears that the larger vocabulary sizes benefit more from the larger number (1000 rather than 500) of random tournaments.

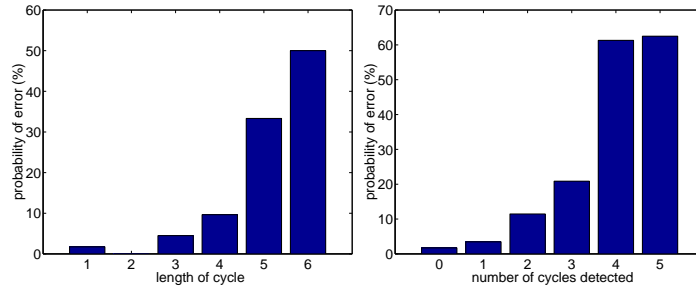

Figure 1: 75-word vocabulary case. Left: probability of error given that there exists a cycle of at least the given length (a cycle length of one means no cycle found). Right:probability of error given that at least the given number of cycles exist.

## 3.1 Empirical Analysis

In order to better understand our results, this section analyzes the 500 and 1000 random tournament strategies described above. Each set of random tournaments produces a set of winners which may be described by a histogram. The entropy $H$ of that histogram describes its spread, and the number of typical winners is approximately $2^H$. This is of course relative to each sample $x$ so we may look at the average ($E[2^H]$), variance, and maximum of this number (the minimum is 1.0 in every case). This is given in Table 3 for the 500 and 1000 cases.

The table indicates that there is typically only one winner since $E2^H$ is approximately 1 and the variances are small. This shows further that the winner is typically not in a cycle, as the existence of a directed cycle in the tournament graph would probably lead to different winners for each random tournament. The relationship between properties of cycles and WER is explored below.

When the tournament is intransitive (and therefore the graph possess a cycle), our second analyses shows that the probability of error tends to increase. This is shown in Figure 1 showing that the error probability increases both as the detected cycle length and the num-

| vocabulary | $L_{ij}$ | skip WER | #cycles(%) | break WER | #cycles(%) |
|---|---|---|---|---|---|
| 75 | 2.33584 | 1.90237 | 13.89 | 1.90223 | 9.34 |
| 150 | 3.31072 | 2.76814 | 19.6625 | 2.67814 | 16.83 |
| 300 | 5.22513 | 4.46296 | 22.38 | 4.46296 | 21.34 |
| 600 | 7.39268 | 6.50117 | 31.96 | 6.50117 | 31.53 |

Table 4: WER results using two strategies (*skip* and *break*) that utilize information about cycles in the tournament graphs, compared to baseline $L_{ij}$. The $4^{th}$ and $6^{th}$ columns show the number of cycles detected relative to the number of samples in each case.

ber of detected cycles increases. [6] This property suggests that the existence of intransitivity could be used as a confidence measure, or could be used to try to reduce errors.

As an attempt at the latter, we evaluated two very simple heuristics that try to eliminate cycles as detected during classification. In the first method (*skip*), we run a sequential tournament (using a random class ordering) until either a clear winner is found (a transitive game), or a cycle is detected. If a cycle is detected, we select two players not in the cycle, effectively jumping out of the cycle, and continue playing until the end of the class ordering. If winner cannot be determined (because there are too few players remaining), we backoff and use $L_{ij}$ to select the winner. In a second method (*break*), if a cycle is detected, we eliminate the class having the smallest likelihood from that cycle, and then continue playing as before. Neither method detects all the cycles in the graph (their number can be exponentially large).

As can be seen, the WER results still provide significant improvements over the baseline, but are no better than earlier results. Because the tournament strategy is coupled with cycle detection, the cycles detected are different in each case (the second method detecting fewer cycles presumably because the eliminated class is in multiple cycles). In any case, it is apparent that further work is needed to investigate the relationship between the existence and properties of cycles and methods to utilize this information.

## 4   Iterative Determination of KL-divergence

In all of our experiments so far, KL-divergence is calculated according to the initial hypothesized answers. We would expect that using the true answers to determine the KL-divergence would improve our results further. The top horizontal lines in Figure 2 shows the original baseline results, and the bottom lines show the results using the true answers (a cheating experiment) to determine the KL-divergence. As can be seen, the improvement is significant thereby confirming that using $D_{ij}$ can significantly improve classification performance. Note also that the relative improvement stays about constant with increasing vocabulary size.

This further indicates that an iterative strategy for determining KL-divergence might further improve our results. In this case, $\hat{L}_{ij}$ is used to determine the answers to compute the first set of KL-divergences used in $S_{ij}^{(1)}$. This is then used to compute a new set of answers which then is used to compute a new scores $S_{ij}^{(2)}$ and so on. The remaining plots in Figure 2 show the results of this strategy for the 500 and 1000 random trials case (i.e., the answers used to compute the KL-divergences in each case are obtained from the previous set of random tournaments using the histogram peak procedure described earlier). Rather surprisingly, the results show that iterating in this fashion does not influence the results in

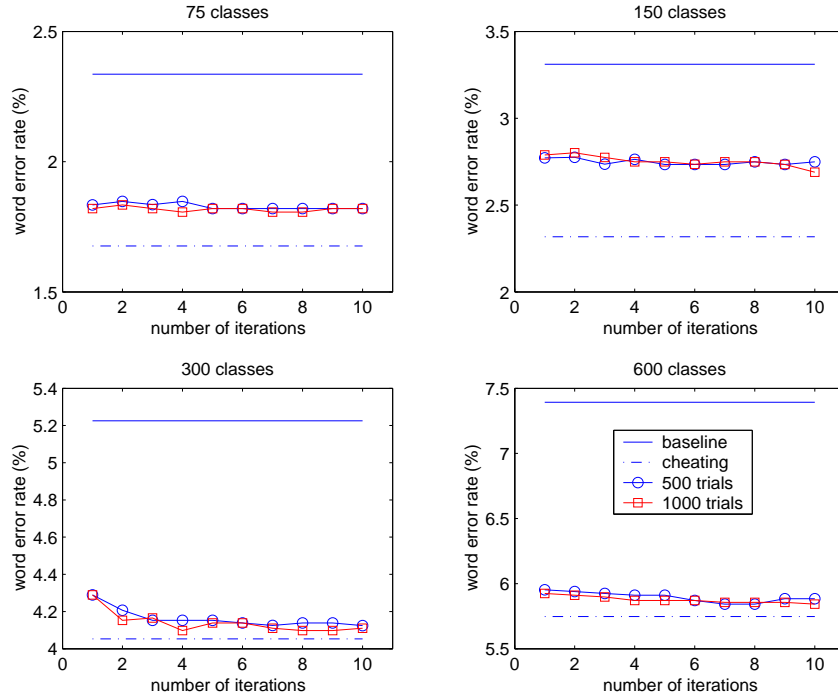

Figure 2: Baseline using likelihood ratio (top lines), cheating results using correct answers for KL-divergence (bottom lines), and the iterative determination of KL-distance using hypothesized answers from previous iteration (middle lines).

any appreciable way — the WERs seem to decrease only slightly from their initial drop. It is the case, however, that as the number of random tournaments increases, the results become closer to the ideal as the vocabulary size increases. We are currently studying further such iterative procedures for recomputing the KL-divergences.

## 5 Discussion and Conclusion

We have introduced a correction term to the likelihood ratio classification method that is justified by the difference between the estimated and true class conditional probabilities $p(x|i), \hat{p}(x|i)$. The correction term $D_{ij}$ is an estimate of the classification bias that would optimally compensate for these differences. The presence of $D_{ij}$ makes the class comparisons intransitive and we introduce several tournament-like strategies to compensate. While the introduction of $D_{ij}$ consistently improves the classification results, further improvements are obtained by the selection of the comparison strategy. Further details and results of our methods will appear in forthcoming publications and technical reports.

## Footnotes

[2]Note that we have dropped the $x$ argument for notational simplicity.

[3]Using 4 state per phone, 12 Gaussian mixtures per state HMMs, totaling 200k free model parameters for the system.

[4]Note, however, that error results here are reported on the development set, i.e., PHONEBOOK lists {a,b,c,d}{o,y}

[5]The 75-word case is an average result of 8 experiments, the 150-word case is an average of 4 cases, and the 300-word case is an average of 2 cases. There are 7291 separate test samples in the 600-word case, and on average about 911 samples per 75-word test case.

[6]Note that this shows a lower bound on the number of cycles detected. This is saying that if we find, for example, four or more cycles then the chance of error is high.

## References

[1] J. Bilmes. *Natural Statistic Models for Automatic Speech Recognition*. PhD thesis, U.C. Berkeley, Dept. of EECS, CS Division, 1999.

[2] T.M. Cover and J.A. Thomas. *Elements of Information Theory*. Wiley, 1991.

[3] R.O. Duda, P.E. Hart, and D.G. Stork. *Pattern Classification*. John Wiley and Sons, Inc., 2000.

[4] J. Pitrelli, C. Fong, S.H. Wong, J.R. Spitz, and H.C. Lueng. PhoneBook: A phonetically-rich isolated-word telephone-speech database. In *Proc. IEEE Intl. Conf. on Acoustics, Speech, and Signal Processing*, 1995.

[5] P.D. Straffin. *Game Theory and Strategy*. The Mathematical ASsociation of America, 1993.
